# Recurrent Networks and NARMA Modeling

**Jerome Connor**  **Les E. Atlas**
FT-10
Interactive Systems Design Laboratory
Dept. of Electrical Engineering
University of Washington
Seattle, Washington 98195

**Douglas R. Martin**
B-317
Dept. of Statistics
University of Washington
Seattle, Washington 98195

## Abstract

There exist large classes of time series, such as those with nonlinear moving average components, that are not well modeled by feedforward networks or linear models, but can be modeled by recurrent networks. We show that recurrent neural networks are a type of nonlinear autoregressive-moving average (NARMA) model. Practical ability will be shown in the results of a competition sponsored by the Puget Sound Power and Light Company, where the recurrent networks gave the best performance on electric load forecasting.

## 1  Introduction

This paper will concentrate on identifying types of time series for which a recurrent network provides a significantly better model, and corresponding prediction, than a feedforward network. Our main interest is in discrete time series that are parsimoniously modeled by a simple recurrent network, but for which, a feedforward neural network is highly non-parsimonious by virtue of requiring an infinite amount of past observations as input to achieve the same accuracy in prediction.

Our approach is to consider predictive neural networks as stochastic models. Section 2 will be devoted to a brief summary of time series theory that will be used to illustrate the the differences between feedforward and recurrent networks. Section 3 will investigate some of the problems associated with nonlinear moving average and state space models of time series. In particular, neural networks will be analyzed as

nonlinear extensions of traditional linear models. From the preceding sections, it will become apparent that the recurrent network will have advantages over feedforward neural networks in much the same way that ARMA models have over autoregressive models for some types of time series.

Finally in section 4, the results of a competition in electric load forecasting sponsored by the Puget Sound Power and Light Company will discussed. In this competition, a recurrent network model gave superior results to feedforward networks and various types of linear models. The advantages of a state space model for multivariate time series will be shown on the Puget Power time series.

## 2    Traditional Approaches to Time Series Analysis

The statistical approach to forecasting involves the construction of stochastic models to predict the value of an observation $x_t$ using previous observations. This is often accomplished using linear stochastic difference equation models, with random inputs.

A very general class of linear models used for forecasting purposes is the class of ARMA(p,q) models

$$x_t = \sum_{i=1}^{p} \phi x_{t-i} + \sum_{j=1}^{q} \theta e_{t-j} + e_t$$

where $e_t$ denotes random noise, independent of past $x_t$. The conditional mean (minimum mean square error) predictor $\hat{x}_t$ of $x_t$ can be expressed in the recurrent form

$$\hat{x}_t = \sum_{i=1}^{p} \phi x_{t-i} + \sum_{j=1}^{q} \theta e_{t-j}.$$

where $e_k$ is approximated by

$$\hat{e}_k = x_k - \hat{x}_k, \quad k = t-1, ..., t-q$$

The key properties of interest for an ARMA(p,q) model are stationarity and invertibility. If the process $x_t$ is stationary, its statistical properties are independent of time. Any stationary ARMA(p,q) process can be written as a moving average

$$x_t = \sum_{k=1}^{\infty} h_k e_{t-k} + e_t.$$

An invertible process can be equivalently expressed in terms of previous observations or residuals. For a process to be invertible, all the poles of the z-transform must lie inside the unit circle of the $z$ plane. An invertible ARMA(p,q) process can be written as an infinite autoregression

$$x_t = \sum_{k=1}^{\infty} \phi_k x_{t-k} + e_t.$$

As an example of how the inverse process occurs, let $e_t$ be solved for in terms of $x_t$ and then substitute previous $e_t$'s into the original process. This can be illustrated

with an MA(1) process

$$x_t = e_t + \theta e_{t-1}$$
$$e_{t-i} = x_{t-i} - \theta e_{t-i-1}$$
$$x_t = e_t + \theta(x_{t-1} - \theta e_{t-2})$$
$$x_t = e_t + \sum_i (-1)^{i-1} \theta^i x_{t-i}$$

Looking at this example, it can be seen that an MA(1) processes with $|\theta| \geq 1$ will depend significantly on observations in the distant past. However, if $|\theta| < 1$, then the effect of the distant past is negligible.

In the nonlinear case, it will be shown that it is not always possible to go back and forth between descriptions in terms of observables (e.g. $x_i$) and descriptions in terms of unobservables (e.g. $e_i$) even when $s_t = 0$. For a review of time series prediction in greater depth see the works of Box [1] or Harvey [2].

## 3  Nonlinear ARMA Models

Many types of nonlinear models have been proposed in the literature. Here we focus on feedforward and recurrent neural networks and how they relate to nonlinear ARMA models.

### 3.1  Nonlinear Autoregressive Models

The simplest generalization to the nonlinear case would be the nonlinear autoregressive (NAR) model

$$x_t = h(x_{t-1}, x_{t-2}, ..., x_{t-p}) + e_t,$$

where $h()$ is an unknown smooth function with the assumption the best (i.e., minimum mean square error) prediction of $x_t$ given $x_{t-1}, ..., x_{t-p}$ is its conditional mean

$$\hat{x}_t = E(x_t | x_{t-1}, ..., x_{t-p}) = h(x_{t-1}, ..., x_{t-p}).$$

Feedforward networks were first proposed as an NAR model for time series prediction by Lapedes and Farber [3]. A feedforward network is a nonlinear approximation to h given by

$$\hat{x}_t = h(x_{t-1}, ..., x_{t-p}) = \sum_{i=1}^{I} W_i f(\sum_{j=1}^{p} w_{ij} x_{t-j}).$$

The weight matrix $w$ is lower diagonal and will allow no feedback. Thus the feedforward network is a nonlinear mapping from previous observation onto predictions of future observations. The function $f(x)$ is a smooth bounded monotonic function, typically a sigmoid.

The parameters $W_i$ and $w_{ij}$ are estimates from a training sample $x_1^0, ..., x_N^0$, thereby obtaining an estimate of $\hat{h}$ of h. Estimates are obtained by minimizing the sum of the square residuals $\sum_{t=1}^{n} (x_t - \hat{x}_t)^2$ by gradient descent procedure known as "backpropagation"[4].

## 3.2  NARMA or NMA

A simple nonlinear generalization of ARMA models is

$$x_t = h(x_{t-1}, x_{t-2}, ..., x_{t-p}, e_{t-1}, ..., e_{t-q}) + e_t.$$

It is natural to predict

$$\hat{x}_t = \hat{h}(x_{t-1}, x_{t-2}, ..., x_{t-p}, \hat{e}_{t-1}, ..., \hat{e}_{t-q}).$$

If the model $\hat{h}(x_{t-1}, x_{t-2}, ..., x_{t-p}, \hat{e}_{t-1}, ..., \hat{e}_{t-q})$ is chosen, then a recurrent network can approximate it as

$$\hat{x}_t = h(x_{t-1}, ..., x_{t-p}) = \sum_{i=1}^{I} W_i f(\sum_{j=1}^{p} w_{ij} x_{t-j} + \sum_{j=1}^{q} w'_{ij}(x_{t-j} - \hat{x}_{t-j})).$$

This model is a special case of the fully interconnected recurrent network

$$\hat{x}_t = \sum_{i=1}^{I} W_i f(\sum_{j=1}^{n} w''_{ij} x_{t-j})$$

where $w''_{ij}$ are coefficients of a full matrix.

Nonlinear autoregressive models and nonlinear moving average models are not always equivalent for nondeterministic processes as in the linear case. If the probability of the next observation depends on the previous state of the process, a representation built on $e_t$ may not be complete unless some information on the previous state is added[8]. The problem is that if $e_t, ..., e_{t-m}$ are known, there is still not enough information to determine which state the series is in at $t - m$. Given the lack of knowledge of the initial state, it is impossible to predict future states and without the state information, the best predictions cannot be made.

If the moving average representation cannot be made with $e_t$ alone, it still may be possible to express a model in terms of past $e_t$ and state information.

$$X_t = h(s_t, s_{t-1}, ..., e_t, e_{t-1}, ....).$$

It has been shown that for a large class of nondeterministic Markov processes, a model of this form can be constructed[8]. This link is important, because a recurrent network is this type of model. For further details on using recurrent networks to NARMA modeling see Connor et al[9].

## 4  Competition on Load Forecasting Data

A fully interconnected recurrent network trained with the Williams and Zipser algorithm [10] was part of a competition to predict the loads of the Puget Sound Power and Light Company from November 11, 1990 to March 31, 1991. The object was to predict the demand for the electric power, known as the load, profile of each day on the previous working day. Because the forecast is made on Friday morning, the Monday prediction is the most difficult. Actual loads and temperatures of the past are available as well as forecasted temperatures for the day of the prediction.

Neural networks are not parsimonious and many parameters need to be determined. Seasonality limits the amount of useful data for the load forecasting problem. For example, the load profile in August is not useful for predicting the load profile in January. This limited amount of data severely constrains the number of parameters a model can accurately determine. We avoided seasonality, while increasing the size of the training set by including data form the last four winters. In total 26976 vectors were available when data from August 1 to March 31 for 1986 to 1990 were included. The larger training set enables neural network models be trained with less danger of overfitting the data. If the network can accurately model load growth over the years, then the network will have the added advantage of being exposed to a larger temperature spectrum on which to base future predictions. The larger temperature spectrum is hypothetically useful for predicting phenomenon such as cold snaps which can result in larger loads than normal. It should be noted that neural networks have been applied to this model in the past[6].

Initially five recurrent models were constructed, one for each day of the week, with Wednesday, Thursday and Friday in a single network. Each network has temperature and load values from a week previous at that hour, the forecasted temperature of the hour to be predicted, the hour year and the week of the forecast. The week of the forecast was included to allow the network to model the seasonality of the data. Some models have added load and temperature from earlier in the week, depending on the availability of the data. The networks themselves consisted of three to four neurons in the hidden layer. This predictor is of the form

$$l_t(k) = \xi_t(k-7) + f(l_t(k-7), \xi_t(k-7), \hat{T}_t(k), T_8(k-1), t, d, y),$$

where $f()$ is a nonlinear function, $l_t(k)$ is the load at time $t$ and day $k$, $\xi_t$ is the noise, $T$ is the temperature, $\hat{T}$ is the forecasted temperature, $d$ is the day of the week, and $y$ is the year of the data.

After comparing its performance to the winner of the competition, the linear model in Fig. 1, the poor performance could be attributed to the choice of model, rather than a problem with recurrent networks. It should be mentioned that the linear model took as one of its inputs, the square of the last available load. This is a parsimonious way of modeling nonlinearities. A second recurrent predictor was then built with the same input and output configuration as the linear model, save the square of the previous load term which the nets nonlinearities can handle. This net, denoted as the Recurrent Network, had a different recurrent model for each hour of the day. Each hour of the day had a different model, this yielded the best predictions. This predictor is of the form

$$l_t(k) = \xi_t(k) + f_t(l_t(k-1), \xi_t(k-1), \hat{T}_t(k), T_8(k-1), d, y).$$

All of the models in the figure use the last available load, forecasted temperature at the hour to be predicted, maximum forecasted temperature of the day to be predicted, the previous midnight temperatures, and the hour and year of the prediction. A second recurrent network was also trained with the last available load at that hour, this enabled $e_{t-1}$ to be modeled. The availability of $e_{t-1}$ turned out to be the difference between making superior and average predictions. It should be noted that the use of $e_{t-1}$ did not improve the results of linear models.

The three most important error measures are the weekly morning, afternoon, and total loads and are listed in the table below. The A.M. peak is the mean average

|  | 83-84 | 84-85 | 85-86 | 90-91 |
|---|---|---|---|---|
| Feedforward | .0180 | .0317 | .0175 | .0331 |
| Recurrent | .0275 | .0355 | .0218 | .0311 |

Table 1: Mean Square Error

percent error (MAPE) of the summed predictions of 7 A.M. to 9 A.M., the P.M. peak is the MAPE of the summed predictions of 5 P.M. to 7 P.M, and the total is the MAPE of the summed predictions over the entire day. Results, of the total power for the day prediction, of the recurrent network and other predictors are shown in Fig. 1. The performance on the A.M. and P.M. peaks were similar[9].

The failure of the daily recurrent network to accurately predict is a product of trying to model to complex a problem. When the complexity of the problem was reduced to that of predicting a single hour of the day, results improved significantly[7].

The superior performance of the recurrent network over the feedforward network is time series dependent. A feedforward and a recurrent network with the same input representation was trained to predict the 5 P.M. load on the previous work day. The feedforward network succeeded in modeling the training set with a mean square error of .0153 compared to the recurrent networks .0179. However, when the tested on several winter outside the training set the results, listed in the table below, varied. For the 1990-91 winter, the recurrent network did better with a mean square error of .0311 compared to the feedforward networks .0331. For the other winter of the years before the training set, the results were quite different, the feedforward network won in all cases. The differences in prediction performance can be explained by the inability of the feedforward network to model load growth in the future. The loads experience in the 1990-91 winter were outside the range of the entire training set. The earlier winters range of loads were not as far form the training set and the feedforward network modeled them well.

The effect of the nonlinear nature of neural networks was apparent in the error residuals of the training and test sets. Figs. 2 and 3 are plots of the residuals against the predicted load for the training and test sets respectively. In Fig. 2, the mean and variance of the residuals is roughly constant as a function of the predicted load, this is indicative of a good fit to the data. However, in Fig. 3, the errors tend to be positive for larger loads and negative for lesser loads. This is a product of the squashing effect of the sigmoidal nonlinearities. The squashing effect becomes acute during the prediction of the peak loads of the winter. These peak loads are caused when a cold spell occurs and the power demand reaches record levels. This is the only measure on which the performance of the recurrent networks is surpassed, human experts outperformed the recurrent network for predictions during cold spells. The recurrent network did outperform all other statistical models on this measure.

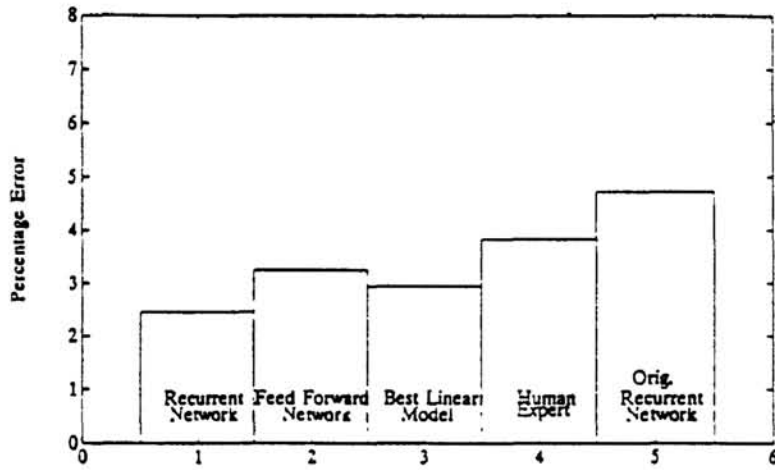

Figure 1: Competition Performance on Total Power

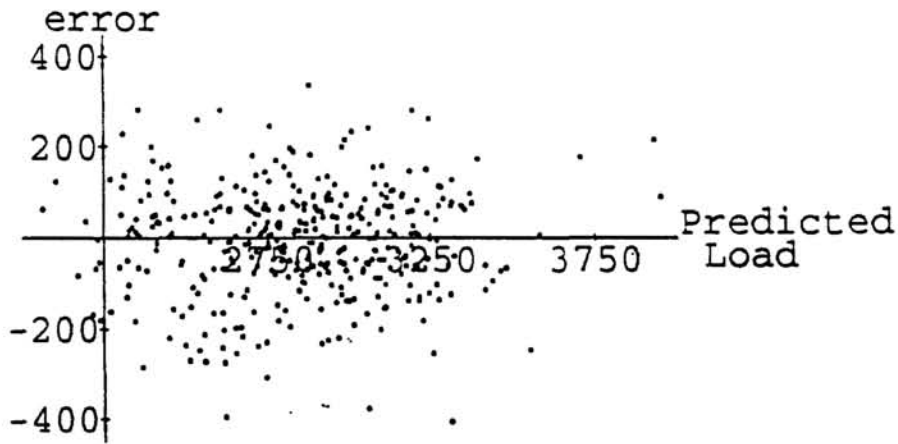

Figure 2: Prediction vs. Residual on Training Set

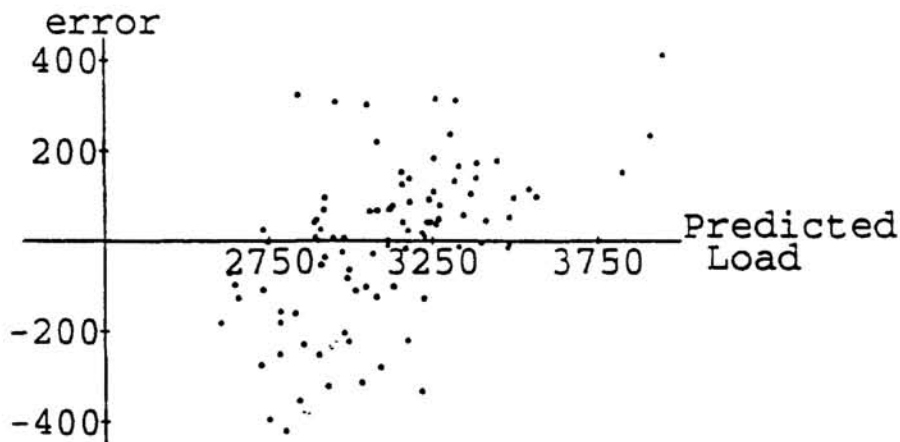

Figure 3: Prediction vs. Residual on Testing Set

## 5   Conclusion

Recurrent networks are the nonlinear neural network analog of linear ARMA models. As such, they are well-suited for time series that possess moving average components, are state dependent, or have trends. Recurrent neural networks can give superior results for load forecasting, but as with linear models, the choice of model is critical to good prediction performance.

## 6   Acknowledgements

We would like to than Milan Casey Brace of the Puget Power Corporation, Dr. Seho Oh, Dr. Mohammed El-Sharkawi, Dr. Robert Marks, and Dr. Mark Damborg for helpful discussions. We would also like to thank the National Science Foundation for partially supporting this work.

## References

[1] G. Box, *Time series analysis: forecasting and control*, Holden-Day, 1976.

[2] A. C. Harvey, *The econometric analysis of time series*, MIT Press, 1990.

[3] A. Lapedes and R. Farber, "Nonlinear Signal Processing Using Neural Networks: Prediction and System Modeling", Technical Report, LA-UR87-2662, Los Alamos National Laboratory, Los Alamos, New Mexico, 1987.

[4] D.E. Rumelhart, G.E. Hinton, and R.J. Williams, "Learning internal representations by error propagation", in *Parallel Distributed Processing*, vol. 1, D.E. Rumelhart, and J.L. NcCelland,eds. Cambridge:M.I.T. Press,1986, pp. 318-362.

[5] M.C. Brace , *A Comparison of the Forecasting Accuracy of Neural Networks with Other Established Techniques*, Proc. of the 1st Int. Forum on Applications of Neural Networks to Power Systems, Seattle, July 23-26, 1991.

[6] L. Atlas, J. Connor, et al.,"Performance Comparisons Between Backpropagation Networks and Classification Trees on Three Real-World Applications", *Advances in Neural Information Processing Systems 2*, pp. 622-629, ed. D. Touretzky, 1989.

[7] S. Oh et al., *Electric Load Forecasting Using an Adaptively Trained Layered Perceptron*, Proc. of the 1st Int. Forum on Applications of Neural Networks to Power Systems, Seattle, July 23-26, 1991.

[8] M. Rosenblatt, *Markov Processes. Structure and Asymptotic Behavior*, Springer-Verlag, 1971, 160-182.

[9] J. Connor, L. E. Atlas, and R. D. Martin,"Recurrent Neural Networks and Time Series Prediction", to be submitted to *IEEE Trans. on Neural Networks*, 1992.

[10] R. Williams and D. Zipser. *A Learning Algorithm for Continually Running Fully Recurrent Neural Networks*, Neural Computation, 1, 1989, 270-280.
